# Fast Feature Selection from Microarray Expression Data via Multiplicative Large Margin Algorithms

**Claudio Gentile**
DICOM, Università dell'Insubria
Via Mazzini, 5, 21100 Varese, Italy
*gentile@dsi.unimi.it*

## Abstract

New feature selection algorithms for linear threshold functions are described which combine backward elimination with an adaptive regularization method. This makes them particularly suitable to the classification of microarray expression data, where the goal is to obtain accurate rules depending on few genes only. Our algorithms are fast and easy to implement, since they center on an incremental (large margin) algorithm which allows us to avoid linear, quadratic or higher-order programming methods. We report on preliminary experiments with five known DNA microarray datasets. These experiments suggest that multiplicative large margin algorithms tend to outperform additive algorithms (such as SVM) on feature selection tasks.

## 1  Introduction

Microarray technology allows researchers to simultaneously measure expression levels associated with thousands or ten thousands of genes in a single experiment (e.g., [7]). However, the number of replicates in these experiments is often seriously limited (tipically a few dozen). This gives rise to datasets having a large number of gene expression values (numerical components) and a relatively small number of samples. As a popular example, in the "Leukemia" dataset from [10] we have only 72 observations of the expression level of 7129 genes. It is clear that in this extreme scenario machine learning methods related to feature selection play a fundamental role for increasing efficiency and enhancing the comprehensibility of the results. Besides, in biological and medical research finding accurate class prediction rules which depend on the level of expression of few genes is important for a number of activities, ranging from medical diagnostics to drug discovery.

Within the classification framework, a *regularization* method (also called penalty-based or feature weighting method) is an indirect route to feature selection. Whereas a (direct) feature selection method searches in the combinatorial space of feature subsets, a regularization method constrains the magnitudes of the parameters assigning them a "degree of relevance" during learning, thereby performing feature selection as a by-product of its learning mechanism (see, e.g., [16, 19, 17, 14, 4, 20]). Feature selection is a wide and active field of research; the reader is referred to [15] for a valuable survey. See also, e.g., [3, 6] (and references therein) for specific work on gene expression data.

In this paper, we introduce novel feature selection algorithms for linear threshold functions,

whose core learning procedure is an incremental large margin algorithm called[1] $\text{ALMA}_p$ (Approximate Large Margin Algorithm w.r.t. norm $p$) [8]. Our $\text{ALMA}_p$-based feature selection algorithms lie between a direct feature selection method and a regularization method. These algorithms might be considered as a refinement on a recently proposed method, specifically tested on microarray expression data, called Recursive Feature Elimination (RFE) [13]. RFE uses Support Vector Machines (SVM) as the core learning algorithm, and performs backward selection to greedily remove the feature whose associated weight is smallest in absolute value until only the desired number of features remain. Our algorithms operate in a similar fashion, but they allow us to eliminate many features at once by exploiting margin information about the current training set. The degree of dimensionality reduction is ruled by the norm $p$ in $\text{ALMA}_p$. The algorithms start by being aggressive (simulating a multiplicative algorithm when the number of current features is large) and end by being gentle (simulating an additive algorithm such as SVM when few features are left). From a computational standpoint, our algorithms lie somewhere between a 1-norm and a 2-norm penalization method. However, unlike other regularization approaches specifically tailored to feature selection, such as those in [4, 20], we do avoid computationally intensive linear (or nonlinear) programming methods. This is because we not only solve the optimization problem associated to regularization in an approximate way, but also use an incremental algorithm having the additional capability to smoothly interpolate between the two kinds of penalizations.

Our algorithms are simple to implement and turn out to be quite fast. We made preliminary experiments on five known DNA microarray datasets. In these experiments, we compared the margin-based feature selection performed by our multiplicative algorithms to a standard correlation-based feature selection method applied to both additive (SVM-like) and multiplicative (Winnow-like) core learning procedures. When possible, we tried to follow previous experimental settings, such as those in [13, 22, 20]. The conclusion of our preliminary study is that a multiplicative (large margin) algorithm is often better that an SVM-like algorithm when the goal is to compute linear threshold rules that are both accurate and depend on the value of few components (as is often the case in gene expression datasets).

## 2 Preliminaries and notation

An *example* is a pair $(\boldsymbol{x}, y)$, where $\boldsymbol{x}$ is an *instance* vector lying in $\mathcal{R}^f$ and $y \in \{-1, +1\}$ is the binary *label* associated with $\boldsymbol{x}$. A *training set* $S$ is a sequence of examples $S = ((\boldsymbol{x}_1, y_1), ..., (\boldsymbol{x}_m, y_m)) \in (\mathcal{R}^f \times \{-1, +1\})^m$. When $F \subseteq \{1, ..., f\}$ is a set of features and $\boldsymbol{v} \in \mathcal{R}^f$, we denote by $\boldsymbol{v}_{|F}$ the subvector of $\boldsymbol{v}$ where the features/dimensions not in $F$ are eliminated. Also, $S_{|F}$ denotes the training set $S_{|F} = ((\boldsymbol{x}_{1|F}, y_1), ..., (\boldsymbol{x}_{m|F}, y_m))$. A weight vector $\boldsymbol{w} = (w_1, ..., w_f) \in \mathcal{R}^f$ represents a hyperplane passing through the origin. As usual, we associate with $\boldsymbol{w}$ the (zero threshold) linear threshold function $\boldsymbol{w} : \boldsymbol{x} \to \text{sign}(\boldsymbol{w} \cdot \boldsymbol{x}) = 1$ if $\boldsymbol{w} \cdot \boldsymbol{x} \geq 0$ and $= -1$ otherwise. When $p \geq 1$ we denote by $||\boldsymbol{w}||_p$ the $p$-norm of $\boldsymbol{w}$, i.e., $||\boldsymbol{w}||_p = (\sum_{i=1}^{f} |w_i|^p)^{1/p}$ (also, $||\boldsymbol{w}||_\infty = \lim_{p\to\infty} (\sum_{i=1}^{f} |w_i|^p)^{1/p}$ $= \max_i |w_i|$). We say that norm $q$ is *dual* to norm $p$ if $q = \frac{p}{p-1}$. In this paper we assume that $p$ and $q$ are some pair of dual values, with $p \geq 2$. We use $p$-norms for instance vectors and $q$-norms for weight vectors. For notational brevity, throughout this paper we use normalized instances $\hat{\boldsymbol{x}} = \boldsymbol{x}/||\boldsymbol{x}||_p$, where $p$ will be clear from the surrounding context. The (normalized) $p$-norm margin (or just the margin) of a hyperplane $\boldsymbol{w}$ with $||\boldsymbol{w}||_q \leq 1$ on example $(\boldsymbol{x}, y)$ is defined as $y\,\boldsymbol{w} \cdot \hat{\boldsymbol{x}}$. If this margin is positive then $\boldsymbol{w}$ classifies $(\boldsymbol{x}, y)$ correctly. Notice that $||\boldsymbol{x}||_p \leq f^{1/p} ||\boldsymbol{x}||_\infty$ for any $\boldsymbol{x} \in \mathcal{R}^f$. Hence if $p$ is logarithmic in the number of features/dimensions of $\boldsymbol{x}$, i.e., $p = \ln f$, we obtain $||\boldsymbol{x}||_{(\ln f)} \leq e\,||\boldsymbol{x}||_\infty$.

<div style="border:1px solid">

**ALGORITHM** $\text{ALMA}_p(S, \alpha)$
**Input:** Training set $S = ((\boldsymbol{x}_1, y_1), ..., (\boldsymbol{x}_m, y_m))$; norm parameter $p \geq 2$; approximation parameter $\alpha \in (0, 1]$.
**Initialization:** $\boldsymbol{w}_1 = \mathbf{0}$; $k = 1$.
**For** $t = 1, 2, ...$ **do**:
Get example $(\boldsymbol{x}_t, y_t)$ and update weights as follows:
Set: $\gamma_k = \frac{\sqrt{8\,(p-1)}}{\alpha}\,\frac{1}{\sqrt{k}}$; $\eta_k = \sqrt{\frac{2}{p-1}}\,\frac{1}{\sqrt{k}}$.
**If** $y_t\,\boldsymbol{w}_k \cdot \hat{\boldsymbol{x}}_t \leq (1 - \alpha)\,\gamma_k$
**then**: $\boldsymbol{w}'_k = \mathbf{T}^{-1}(\mathbf{T}(\boldsymbol{w}_k) + \eta_k\,y_t\,\hat{\boldsymbol{x}}_t)$,
$\boldsymbol{w}_{k+1} = \boldsymbol{w}'_k / ||\boldsymbol{w}'_k||_q$, where $q = \frac{p}{p-1}$,
$k \leftarrow k + 1$.
**Output:** Final weight vector $\boldsymbol{w}_k = (w_{k,1}, ..., w_{k,f})$; final margin $\gamma = \gamma_k$.

</div>

Figure 1: The approximate large margin algorithm $\text{ALMA}_p$.

Also, $||\boldsymbol{w}||_1 \leq 1$ implies $||\boldsymbol{w}||_q \leq 1$ for any $q > 1$. Thus if $||\boldsymbol{w}||_1 \leq 1$ the $(\ln f)$-norm margin $\frac{y\,\boldsymbol{w}\cdot\boldsymbol{x}}{||\boldsymbol{x}||_{(\ln f)}}$ is actually bounded from below by the $\infty$-norm margin $\frac{y\,\boldsymbol{w}\cdot\boldsymbol{x}}{||\boldsymbol{x}||_\infty}$ divided by some constant. Arguing about the $\infty$-norm margin is convenient when dealing with *sparse* hyperplanes, i.e., with hyperplanes having only a small number of relevant features (e.g., [14]). We say that a training set $S = ((\boldsymbol{x}_1, y_1), ..., (\boldsymbol{x}_m, y_m))$ is linearly separable with margin $\gamma > 0$ when there exists a hyperplane $\boldsymbol{w}$ with $||\boldsymbol{w}||_q \leq 1$ such that $y_t\,\boldsymbol{w}\cdot\hat{\boldsymbol{x}}_t \geq \gamma$ for $t = 1, ..., m$. Given $\alpha \in (0, 1]$, we say that hyperplane $\boldsymbol{w}'$ is an $\alpha$-*approximation* to $\boldsymbol{w}$ (w.r.t. training set $S$) if $||\boldsymbol{w}'||_q \leq 1$ and $y_t\,\boldsymbol{w}'\cdot\hat{\boldsymbol{x}}_t \geq (1-\alpha)\gamma$ holds for $t = 1, ..., m$. In particular, if the underlying margin is an $\infty$-norm margin (and $\alpha$ is not close to 1) then $\boldsymbol{w}'$ tends to share the sparsity properties of $\boldsymbol{w}$. See also Section 3.

## 3 The large margin algorithm $\text{ALMA}_p$

$\text{ALMA}_p$ is a large margin variant of the $p$-norm Perceptron algorithm[2] introduced by [11] (see also [9]). The version of the algorithm we have used in our experiments is described in Figure 1, where the one-one mapping $\mathbf{T} = (T_1, ..., T_f) : \mathcal{R}^f \to \mathcal{R}^f$ is the gradient of the scalar function $\frac{1}{2}||\cdot||_q^2$ and its inverse $\mathbf{T}^{-1} = (T_1^{-1}, ..., T_f^{-1}) : \mathcal{R}^f \to \mathcal{R}^f$ is the gradient of the (Legendre dual) function $\frac{1}{2}||\cdot||_p^2$. The mapping $\mathbf{T}$ depends on the chosen norm $p$, which we omit for notational brevity. One can immediately see that $p = q = 2$ gives $\mathbf{T} = \mathbf{T}^{-1}$ = identity. See [9] for further discussion about the properties of $\mathbf{T}$. The algorithm in Figure 1 takes in input a training set $S = ((\boldsymbol{x}_1, y_1), ..., (\boldsymbol{x}_m, y_m)) \in (\mathcal{R}^f \times \{-1, +1\})^m$, a norm value $p \geq 2$ and a parameter $\alpha \in (0, 1]$, measuring the degree of approximation to the optimal margin hyperplane. Learning proceeds in a sequence of trials. $\text{ALMA}_p$ maintains a normalized vector $\boldsymbol{w}_k$ of $f$ weights. It starts from $\boldsymbol{w}_1 = \mathbf{0}$ and in the generic trial $t$ it processes example $(\boldsymbol{x}_t, y_t)$. If the current weight vector $\boldsymbol{w}_k$ classifies $(\boldsymbol{x}_t, y_t)$ with (normalized) margin not larger than $(1-\alpha)\,\gamma_k$ then the algorithm updates its internal state. The update rule consists of the following: First, the algorithm computes $\boldsymbol{w}'_k$ via a ($p$-norm) perceptron-like update rule. Second, $\boldsymbol{w}'_k$ is normalized w.r.t. the chosen norm $q$ (recall that $q$ is dual to $p$). The normalized vector $\boldsymbol{w}_{k+1}$ will then be used in the next trial. After sweeping (typically more than once) through the training set, the algorithm outputs an $f$-dimensional vector $\boldsymbol{w}_k$ which represents the linear model the algorithm has learned from the data. The output also includes the final margin $\gamma = \gamma_k$, where $k$ is the total number of updates (plus one) the algorithm took to compute $\boldsymbol{w}_k$. This margin is a valuable indication of the level of "noise" in the data. In particular, when the training set $S$ is linearly separable,

we can use $\gamma$ to estimate from above the true margin $\gamma^*$ of $S$ (see Theorem 1). In turn, $\gamma^*$ helps us in setting up a reliable feature selection process (see Section 4). Theorem 1 is a convergence result stating two things [8]: 1. $\text{ALMA}_p(S, \alpha)$ computes an $\alpha$-approximation to the maximal $p$-norm margin hyperplane after a finite number of updates; 2. the margin $\gamma$ output by $\text{ALMA}_p(S, \alpha)$ is an upper bound on the true margin of $S$.[3]

**Theorem 1** *[8] Let* $\gamma^* = \max_{\boldsymbol{w} \in \mathcal{R}^f : ||\boldsymbol{w}||_q = 1} \min_{t=1,...,m} y_t \, \boldsymbol{w} \cdot \hat{\boldsymbol{x}}_t > 0$. *Then the number of updates made by the algorithm in Figure 1 (i.e., the number of trials $t$ such that $y_t \, \boldsymbol{w}_k \cdot \hat{\boldsymbol{x}}_t \leq (1 - \alpha) \, \gamma_k$) is upper bounded by* $\frac{2\,(p-1)}{(\gamma^*)^2} \left( \frac{2}{\alpha} - 1 \right)^2 + \frac{8}{\alpha} - 4 = O\left( \frac{p-1}{\alpha^2\,(\gamma^*)^2} \right)$. *Furthermore, throughout the run of the algorithm we have $\gamma_k \geq \gamma \geq \gamma^*$, for $k = 1, 2, ...$ (recall that $\gamma$ is the last $\gamma_k$ produced by $\text{ALMA}_p$). Hence the previous bound is also an upper bound on the number of trials $t$ such that $y_t \, \boldsymbol{w}_k \cdot \hat{\boldsymbol{x}}_t \leq (1 - \alpha) \, \gamma$.*

Recalling Section 2, we notice that setting $p = O(\ln f)$ makes $\text{ALMA}_p$ useful when learning sparse hyperplanes. In particular, the above theorem gives us the following $\infty$-norm margin upper bound on the number of updates: $O\left( \ln f \, / \, (\alpha^2\,(\gamma^*)^2) \right)$, where $\gamma^* = \max_{\boldsymbol{w} \in \mathcal{R}^f : ||\boldsymbol{w}||_1 = 1} \min_{t=1,...,m} y_t \, \boldsymbol{w} \cdot \boldsymbol{x}_t \, / \, ||\boldsymbol{x}_t||_\infty$. This is similar to the behavior exhibited by classifiers based on linear programming (e.g., [17, 19, 4] and references therein), as well as to the performance achieved by multiplicative algorithms, such as the zero-threshold Winnow algorithm [11].

## 4 The multiplicative feature selection algorithms

We now describe two feature selection algorithms based on $\text{ALMA}_p$. The algorithms differ in the way features are eliminated. The first algorithm, called ALMA-FS (ALMA-based Feature Selection), is strongly influenced by its training behavior: If $\text{ALMA}_p$ has made many updates during training then arguably this corresponds to a high level of noise in the data (w.r.t. a linear model). In this case the feature selection mechanism tends to be prudent in eliminating features. On the other hand, if the number of updates is small we can think of the linear model computed by $\text{ALMA}_p$ as an accurate one for the training data at hand, so that one can reliably perform a more aggressive feature removal. The second algorithm, called $\text{ALMA}_{\ln}$-RFE, performs Recursive Feature Elimination (RFE) on the linear model computed by $\text{ALMA}_p$, and might be seen as a simplified version of the first one, where the rate of feature removal is constant and the final number of features is fixed ahead of time. ALMA-FS is described in Figure 2. It takes in input a training set $S = ((\boldsymbol{x}_1, y_1), ..., (\boldsymbol{x}_m, y_m)) \in (\mathcal{R}^n \times \{-1, +1\})^m$ and a parameter $\alpha$ (which is the same as $\text{ALMA}_p$'s). Then the algorithm repeatedly invokes $\text{ALMA}_p$ on the same training set but progressively reducing the set $F$ of current features. The algorithm starts with $F = \{1, ..., n\}$, being $n$ the dimension of the input space. Then, on each repeat-until iteration, the algorithm: sets the norm $p$ to the logarithm[4] of the number $f$ of current features, runs $\text{ALMA}_p$ for the given values of $\alpha$ and $p$, gets in output $\boldsymbol{w}$ and $\gamma$, and computes the new (smaller) $F$ to be used in the next iteration. Computing the new $F$ amounts to sorting the components of $\boldsymbol{w}$ according to decreasing absolute value and then keeping, among the $f$ features, only the largest ones (thereby eliminating features which are likely to be irrelevant). Here $c(\alpha) \in [0, 1]$ is a suitable function whose value will be specified later. We call a repeat-until iteration of this kind a feature selection *stage*. ALMA-FS terminates when it reaches a local minimum $F$, where the algorithm is unable to drop any further features.

ALMA-FS uses the output produced by $\text{ALMA}_p$ in the most natural way, retaining only the features corresponding to (supposedly) relevant components of $\boldsymbol{w}$. We point out that here the discrimination between relevant and irrelevant components is based on the margin $\gamma$

**ALGORITHM** ALMA-FS$(S, \alpha)$
**Input:** Training set $S = ((\boldsymbol{x}_1, y_1), ..., (\boldsymbol{x}_m, y_m))$; approx. param. $\alpha \in (0, 1]$.
**Initialization:** $F = \{1, 2, ..., n\}$; $f := |F| = n$.
**Repeat**

- Set $p := \max\{2, \ln f\}$ and run ALMA$_p(S_{|F}, \alpha)$, getting in output $\boldsymbol{w} = (w_1, ..., w_f) \in \mathcal{R}^f$ and $\gamma > 0$;
- Sort $w_1, ..., w_f$ according to decreasing $|w_i|$ and let $w_{i_1}, ..., w_{i_f}$ be the sorted sequence; set $q = \frac{p}{p-1}$ and compute the smallest $f^* \leq f$ s.t.

$$\sum_{j=1}^{f^*} |w_{i_j}|^q \geq 1 - (c(\alpha)\,\gamma)^q; \qquad (1)$$

- Set $F = \{i_1, i_2, ..., i_{f^*}\}$; $f := |F| = f^*$;

**Until** $F$ does not shrink any more.
**Output:** Final weight vector $\boldsymbol{w} = (w_1, ..., w_f)$.

---

Figure 2: ALMA-FS: Feature selection using ALMA$_p$ where $p$ is logarithmic in $f$.

output by ALMA$_p$. In turn, $\gamma$ depends on the number of training updates made by ALMA$_p$, i.e., on the "amount of noise" in the data. This criterion can be viewed as a margin-based criterion according to the following fact: If in any given stage ALMA$_p$ has computed an $\alpha$-approximation to the maximal margin hyperplane for a (linearly separable) training sequence $S$, then the (smaller) vector computed at the end of that stage will be an $(\alpha + c(\alpha))$-approximation to the maximal margin hyperplane for the new (linearly separable) sequence where some features have been eliminated. This statement follows directly from (1) and Theorem 1. We omit the details due to space limitations. From this point of view, a reasonable choice of $c(\alpha)$ is one which insures $\alpha + c(\alpha) \leq 1$ for $\alpha \in [0, 1]$ and the two limiting conditions $\lim_{\alpha \to 0} \alpha + c(\alpha) = 0$ and $\lim_{\alpha \to 1} \alpha + c(\alpha) = 1$. The simplest function satisfying the conditions above (the one we used in the experiments) is $c(\alpha) = \alpha\,(1 - \alpha)$. ALMA-FS starts with a relatively large value of the norm parameter $p$ (making it fairly aggressive at the beginning), and then progressively reduces this parameter so that the algorithm can focus in later stages on the remaining features. This heuristic approach allows us to keep a good approximation capability (as measured by the margin) while dropping a lot of irrelevant components from the weight vectors computed by ALMA$_p$.

ALMA$_{\ln}$-RFE is a simplified version of ALMA-FS that halves the number of features in each stage, and uses again a norm $p$ logarithmic in the number of current features. The $\alpha$ parameter is replaced by $n_f$, the desired number of features. ALMA$_{\ln}$-RFE$(S, n_f)$ is obtained from the algorithm in Figure 2 upon replacing the definition of $f^*$ in (1) by $f^* = \max\{\lfloor f/2 \rfloor, n_f\}$, so that the number of training stages is always logarithmic in $n/n_f$.

## 5 Experiments

We tested ALMA-FS and ALMA$_{\ln}$-RFE on a few well-known microarray datasets (see below). For the sake of comparison, we tended to follow previous experimental settings, such as those described in [13, 22, 20]. Our results are summarized in Table 1. For each dataset, we first generated a number of random training/test splits. Since we used on-line algorithms, the output depends on the order of the training sequence. Therefore our random splits also included random permutations of the training set. The results shown in Table 1 are averaged over these random splits.

Five datasets have been used in our experiments.

1. The ALL-AML dataset [10] contains 72 samples, each with expression profiles about 7129 genes. The task is to distinguish between the two variants of leukemia ALL and AML. We call this dataset the "Leukemia" dataset. We used the first 38 examples as training set and the remaining 34 as test set. This seems to be a standard training/test split (e.g., [10, 21, 13, 22]). The results have been averaged over 1000 random permutations of the

training set.

2. The "Colon Cancer" dataset [2] contains 62 expression profiles for tumor and normal samples concerning 2000 genes. Following [20], we randomly split the dataset into a training set of 50 examples and a test set of 12. The random split was performed 1000 times.

3. In the ER+/ER− dataset from [12] the task is to analyze expression profiles of breast cancer and classify breast tumors according to ER (Estrogen Receptor) status. This dataset (which we call the "Breast" dataset) contains 58 expression profiles concerning 3389 genes. We randomly split 1000 times into a training set of size 47 and a test set of size 11.

4. The "Prostate" cancer dataset from [18] contains 102 samples with expression profiles concerning 12600 genes. The task is to separate tumor from normal samples. As in [18], we estimated the test error through a Leave-One Out Cross Validation (LOOCV)-like estimator. In particular, for this dataset we randomly split 1000 times into a training set of 101 examples and a test set of 1 example, and then averaged the results. (This is roughly equivalent to LOOCV with 10 random permutations of the training set.)

5. In the "Lymphoma" dataset [1] the goal is to separate cancerous and normal tissues in a large B-Cell lymphoma problem. The dataset contains 96 expression profiles concerning 4026 genes, 62 samples are in the classes "DLCL", "FL" and "CLL" (malignant) and the remaining 34 are labelled "otherwise". As in [20], we randomly split the dataset into a training set of size 60 and a test set of size 36. The random split was performed 1000 times.

We made no preprocessing on the data. All our experiments have been run on a PC with a single AMD Athlon processor running at 1300 Mhz. The running times we will be giving are measured on this machine. We compared on these datasets ALMA-FS ("FS" in Table 1) and $\text{ALMA}_{\ln}$-RFE ("ln-RFE") to three more feature selection algorithms: a fast approximation to Recursive Feature Elimination applied to SVM (called $\text{ALMA}_2$-RFE, abbreviated as "2-RFE" in Table 1), and a standard feature selection method based on correlation coefficients (e.g., [10]) applied to both (an approximation to) SVM and $\text{ALMA}_{\ln f}$, being $f$ the number of features selected by the correlation method. We call the last two methods $\text{ALMA}_2$-CORR ("2-CORR" in Table 1) and $\text{ALMA}_{\ln}$-CORR ("ln-CORR" in Table 1), respectively. In all cases our base learning algorithm was $\text{ALMA}_p(.,\alpha)$, where $\alpha \in \{0.5, 0.6, 0.7, 0.8, 0.9\}$, and $p$ was either 2 (to approximate SVM) or logarithmic in the number of features the algorithm was operating on (to simulate a multiplicative large margin algorithm). For each combination (algorithm, number of genes), only the best accuracy results (w.r.t. $\alpha$) are shown. On the "Colon cancer", the "Breast" and the "Lymphoma" datasets we run $\text{ALMA}_p$ by cycling 50 times over the current training set. On the "Leukemia" and the "Prostate" datasets (which are larger) we cycled 100 times. In Table 1 we give, for each dataset, the average error and the number of features ("# GENES") selected by the algorithms.[5] The only algorithm which tries to determine the final number of features as a part of its inference mechanism is ALMA-FS: all the others take this number as an explicit input parameter.

The main goal of this experimental study was to carry out a direct comparison between different feature selection methods combined with different core learning algorithms. Feature selection performed by ALMA-FS, $\text{ALMA}_{\ln}$-RFE and $\text{ALMA}_2$-RFE is margin-based, while feature selection performed by $\text{ALMA}_2$-CORR and $\text{ALMA}_{\ln}$-CORR is correlation-based. According to [15], the former falls within the category of *wrapper* methods, while the latter is an example of *filter* methods. The two core learning algorithms we employed are the SVM-like algorithm $\text{ALMA}_2$ and the (large margin) Winnow-like algorithm $\text{ALMA}_p$, with logarithmic $p$. The first has been used with $\text{ALMA}_2$-RFE and $\text{ALMA}_2$-CORR, the second has been used with ALMA-FS, $\text{ALMA}_{\ln}$-RFE and $\text{ALMA}_{\ln}$-CORR.

The accuracy results we have obtained are often superior to those reported in the litera-

Table 1: Experimental results on five microarray datasets. The percentages denote the average fraction of misclassified patterns in the test set, while "# GENES" denotes the average number of genes (features) selected. The results refer to the same training/test splits. Notice that ALMA-FS ("FS") determines automatically the number of genes to select. According to Wilcoxon signed rank test, $\geq 0.5\%$ accuracy difference might be considered significant.

|  | # GENES | FS | 2-RFE | ln-RFE | 2-CORR | ln-CORR |
|---|---|---|---|---|---|---|
| LEUKEMIA | 20 | — | 5.8% | 3.3% | 5.9% | 3.7% |
|  | 26.5 | 3.0% | — | — | — | — |
|  | 40 | — | 6.7% | 3.0% | 5.0% | 3.6% |
|  | 60 | — | 8.9% | 3.2% | 4.3% | 2.9% |
|  | 100 | — | 9.0% | **2.5%** | 4.0% | 2.9% |
|  | 200 | — | 7.2% | 3.1% | 3.0% | 4.5% |
|  | ALL | — | 3.5% | 3.3% | 3.5% | 3.3% |
| COLON CANCER | 20 | — | 17.0% | 13.1% | 15.4% | 14.8% |
|  | 22.6 | 12.7% | — | — | — | — |
|  | 40 | — | 15.4% | 12.1% | 14.4% | 14.0% |
|  | 60 | — | 14.8% | **12.0%** | 14.2% | 13.6% |
|  | 100 | — | 14.3% | 12.6% | 13.7% | 13.1% |
|  | 200 | — | 13.2% | 12.4% | 13.9% | 13.2% |
|  | ALL | — | 13.0% | 13.3% | 13.0% | 13.3% |
| BREAST | 20 | — | 11.5% | 10.3% | 6.1% | **5.5%** |
|  | 38.5 | 9.5% | — | — | — | — |
|  | 40 | — | 10.7% | 9.9% | 6.5% | 6.5% |
|  | 60 | — | 10.1% | 9.9% | 7.5% | 8.5% |
|  | 100 | — | 10.4% | 9.8% | 13.1% | 10.4% |
|  | 200 | — | 11.9% | 9.6% | 14.6% | 14.5% |
|  | ALL | — | 15.8% | 10.0% | 15.8% | 10.0% |
| PROSTATE | 20 | — | 8.4% | 7.8% | 11.5% | 10.4% |
|  | 30.8 | 9.5% | — | — | — | — |
|  | 40 | — | 8.1% | 9.4% | 10.2% | 8.0% |
|  | 60 | — | 8.1% | 10.3% | 8.5% | 7.7% |
|  | 100 | — | 9.3% | 10.2% | 6.9% | **6.5%** |
|  | 200 | — | 9.8% | 9.9% | 8.4% | 7.2% |
|  | ALL | — | 10.0% | 10.4% | 10.0% | 10.4% |
| LYMPHOMA | 20 | — | 10.1% | 9.9% | 12.6% | 12.3% |
|  | 30.8 | 8.1% | — | — | — | — |
|  | 40 | — | 7.9% | 7.4% | 10.5% | 10.2% |
|  | 60 | — | 7.4% | 6.8% | 9.5% | 9.2% |
|  | 100 | — | 6.6% | 6.0% | 8.2% | 8.3% |
|  | 200 | — | 6.3% | 5.6% | 7.4% | 7.7% |
|  | ALL | — | 7.2% | **5.5%** | 7.2% | **5.5%** |

ture, though this should not be considered very significant.[6] From our direct comparison, however, a few (more reliable) conclusions can be drawn. First, on these gene expression

datasets a large margin Winnow-like algorithm generally outperforms an SVM-like algorithm. Second, despite the common wisdom [15] according to which wrapper methods tend to be more accurate than filter methods, it is hard to tell here how the two methods compare (see [22] for similar results). Third, knowing the "optimal" number of genes beforehand is a valuable side information. Notice that, unlike many of the methods proposed in the literature, ALMA-FS tries to determine in an automatic way a "good" number of features to select.[7] In fact, due to the scarcity of examples and the large number of vector components, the repeated use of cross-validation on the same validation set might lead to overfitting. ALMA-FS seems to do a fine job of it on three out of five datasets (on the "Breast" dataset "FS" should only be compared to "2-RFE" and "ln-RFE"). Finally, we would like to stress that our feature selection algorithms are quite fast. To give an idea, on the "Colon Cancer" and the "Breast" datasets our algorithms take on average just a few seconds, while on the "Prostate" dataset they take just a few minutes.

## Footnotes

[1]Broadly speaking, as the norm parameter $p$ is varied, $\text{ALMA}_p$ is able to (approximately) interpolate between Support Vector Machines [5] and (large margin versions of) multiplicative classification algorithms, such as Winnow [16]. Compared to Winnow, $\text{ALMA}_p$ is more flexible (since we can adjust the norm parameter $p$) and requires less tuning. See Section 3 for details.

[2]The $p$-norm Perceptron algorithm is a generalization of the classical Perceptron algorithm, obtained by setting $p = 2$.

[3] A more general statement holds for the nonseparable case (see [8] for details). In this case, the $\alpha$ parameter in $\text{ALMA}_p(., \alpha)$ is similar to the $C$ parameter in SVM [5].

[4] In order to prevent $p < 2$, we actually set $p = 2$ when $\ln f < 2$.

[5]Observe that, due to the on-line nature of the algorithms, different sets of genes get selected on different runs. Therefore one could also collect statistics about the gene selection frequency over the runs. Details will be given in the full paper.

[6]In fact, the results on feature selection applied to microarray datasets are not readily comparable across different papers, due to the randomness in the training/test splits (which is a relevant source of variance) and the different preprocessing of the data. That said, we briefly mention a few results reported by other researchers on the same datasets. On the "Leukemia" dataset, [22] report 0% test error for a logistic regression algorithm that chooses the number of features to extract by LOOCV. The same error rate is reported by [21] for a linear SVM using 20 genes. [20] use linear SVM as the underlying learning algorithm. On the "Colon Cancer" dataset, the authors report an average accuracy of 16.4% without feature selection and an accuracy ranging between 15.0% and 16.9% (depending on the number of genes selected) for the RFE and the AROM (Approximation of the Zero-Norm Minimization) methods. On the "Lymphoma" dataset the same authors report 7.1% average error for linear SVM and 5.9% to 6.8% average error (again depending on the number of genes selected) for the RFE and the AROM methods. On the "Prostate" dataset, [18] use a $k$-NN classifier and report a LOOCV accuracy comparable to ALMA$_2$-RFE's (but worse than ALMA$_{ln}$-CORR's).

[7]The reader might object that the number of selected features can depend on the value of parameter $\alpha$ in ALMA$_p$. In practice, however, we observed that $\alpha$ does not have a big influence on this number.

# References

[1] Alizadeh, A., et al. (2000). Distinct types of diffuse large b-cell lymphoma identified by gene expression profiling. *Nature*, *403*, 503–511.

[2] Alon, U., et al. (1999). Broad patterns of gene expression revealed by clustering analysis of tumor and normal colon cancer tissues probed by oligonucleotide arrays. *Cell Biol.*, *96*, 6745–6750.

[3] Ben-Dor, A., et al. (2000). Tissue classification with gene expression profiles. *J. Comput. Biol.*, 7, 559–584.

[4] Bradley, P., & Mangasarian, O. (1998). Feature selection via concave minimization and support vector machines. *Proc. 15th ICML* (pp. 82–90).

[5] Cortes, C., & Vapnik, V. (1995). Support-vector networks. *Machine Learning*, *20(3)*, 273–297.

[6] Dudoit, S., Fridlyand, J., & Speed T.P. (2002). Comparison of discrimination methods for the classification of tumors using gene expression data. *JASA*, 97(457), 77–87.

[7] Fodor, S. (1997). Massively parallel genomics. *Science*, *277*, 393–395.

[8] Gentile, C. (2001a). A new approximate maximal margin classification algorithm. *JMLR*, *2*, 213–242.

[9] Gentile, C. (2001b). The robustness of the $p$-norm algorithms. *Machine Learning J., to appear*.

[10] Golub, T., et al. (1999). Molecular classification of cancer: Class discovery and class prediction by gene expression. *Science*, *286*, 531–537.

[11] Grove, A., Littlestone, N., & Schuurmans, D. (2001). General convergence results for linear discriminant updates. *Machine Learning Journal*, *43(3)*, 173–210.

[12] Gruvberger, S., et al. (2001). Estrogen receptor status in breast cancer is associated with remarkably distinct gene expression patterns. *Cancer Res.*, *61*, 5979–5984.

[13] Guyon, I., Weston, J., Barnhill, S., & Vapnik, V. (2002). Gene selection for cancer classification using support vector machines. *Machine Learning Journal*, *46(1-3)*, 389–422.

[14] Kivinen, J., Warmuth, M., & Auer, P. (1997). The perceptron algorithm vs. winnow: linear vs. logarithmic mistake bounds when few input variables are relevant. *AI*, *97*, 325–343.

[15] Kohavi, R., & John, G. (1997). Wrappers for feature subset selection. *AI*, *97*, 273–324.

[16] Littlestone, N. (1988). Learning quickly when irrelevant attributes abound: A new linear-threshold algorithm. *Machine Learning*, *2*, 285–318.

[17] Mangasarian, O. (1997). Mathematical programming in data mining. *DMKD*, *42(1)*, 183–201.

[18] Singh, D., et al. (2002). Gene expression correlates of clinical prostate cancer behavior. *Cancer Cell*, *1*.

[19] Tibshirani, R. (1995). Regression selection and shrinkage via the lasso. *JRSS B*, *1*, 267–288.

[20] Weston, J., Elisseeff, A., Scholkopf, B., & Tipping, M. (2002). The use of zero-norm with linear models and kernel methods. *JMLR, to appear*.

[21] Weston, J., Mukherjee, S., Chapelle, O., Pontil, M., Poggio, T., & Vapnik, V. (2000). Feature selection for svms. *Proc. NIPS 13*.

[22] Xing, E., Jordan, M., & Karp, R. (2001). Feature selection for high-dimensional genomic microarray data. *Proc. 18th ICML*.

